# Robust, Efficient, Globally-Optimized Reinforcement Learning with the Parti-Game Algorithm

**Mohammad A. Al-Ansari** and **Ronald J. Williams**
College of Computer Science, 161 CN
Northeastern University
Boston, MA 02115
alansar@ccs.neu.edu, rjw@ccs.neu.edu

## Abstract

Parti-game (Moore 1994a; Moore 1994b; Moore and Atkeson 1995) is a reinforcement learning (RL) algorithm that has a lot of promise in overcoming the curse of dimensionality that can plague RL algorithms when applied to high-dimensional problems. In this paper we introduce modifications to the algorithm that further improve its performance and robustness. In addition, while parti-game solutions can be improved locally by standard local path-improvement techniques, we introduce an add-on algorithm in the same spirit as parti-game that instead tries to improve solutions in a non-local manner.

## 1 INTRODUCTION

Parti-game operates on goal problems by dynamically partitioning the space into hyper-rectangular cells of varying sizes, represented using a k-d tree data structure. It assumes the existence of a pre-specified local controller that can be commanded to proceed from the current state to a given state. The algorithm uses a game-theoretic approach to assign costs to cells based on past experiences using a minimax algorithm. A cell's cost can be either a finite positive integer or infinity. The former represents the number of cells that have to be traveled through to get to the goal cell and the latter represents the belief that there is no reliable way of getting from that cell to the goal. Cells with a cost of infinity are called *losing* cells while others are called *winning* ones.

The algorithm starts out with one cell representing the entire space and another, contained within it, representing the goal region. In a typical step, the local controller is commanded to proceed to the center of the most promising neighboring cell. Upon entering a neighboring cell (whether the one aimed at or not), or upon failing to leave the current cell within

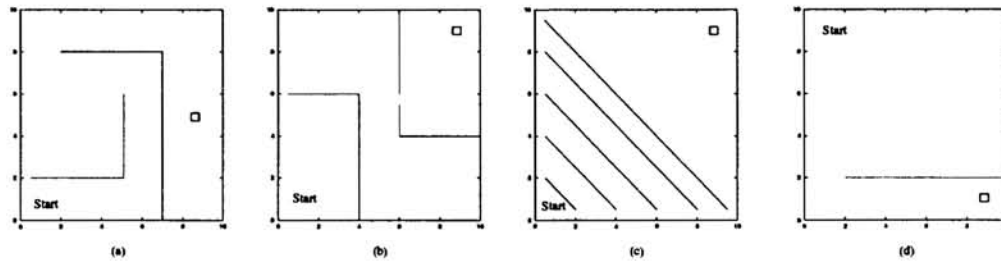

Figure 1: In these mazes, the agent is required to start from the point marked Start and reach the square goal cell.

a timeout period, the result of this attempt is added to the database of experiences the algorithm has collected, cell costs are recomputed based on the updated database, and the process repeats. The costs are computed using a Dijkstra-like, one-pass minimax version of dynamic programming. The algorithm terminates upon entering the goal cell.

If at any point the algorithm determines that it can not proceed because the agent is in a losing cell, each cell lying on the boundary between losing and winning cells is split across the dimension in which it is largest and all experiences involving cells that are split are discarded. Since parti-game assumes, in the absence of evidence to the contrary, that from any given cell every neighboring cell is reachable, discarding experiences in this way encourages exploration of the newly created cells.

## 2   PARTITIONING ONLY LOSING CELLS

The win-lose boundary mentioned above represents a barrier the algorithm perceives that is preventing the agent from reaching the goal. The reason behind partitioning cells along this boundary is to increase the resolution along these areas that are crucial to reaching the goal and thus creating more regions along this boundary for the agent to try to get through. By partitioning on both sides of the boundary, parti-game guarantees that neighboring cells along the boundary remain close in size. Along with the strategy of aiming towards centers of neighboring cells, this produces pairings of winner-loser cells that form proposed "corridors" for the agent to try to go through to penetrate the barrier it perceives.

In this section we investigate doing away with partitioning on the winning side, and only partition losing cells. Because partitioning can only be triggered with the agent on the losing side of the win-lose boundary, partitioning only losing cells would still give the agent the same kind of access to the boundary through the newly formed cells. However, this would result in a size disparity between winner- and loser-side cells and, thus, would not produce the winner side of the pairings mentioned above. To produce a similar effect to the pairings of parti-game, we change the aiming strategy of the algorithm. Under the new strategy, when the agent decides to go from the cell it currently occupies to a neighboring one, it aims towards the center point of the common surface between the two cells. While this does not reproduce the same line of motion of the original aiming strategy exactly, it achieves a very similar objective.

Parti-game's success in high-dimensional problems stems from its variable resolution strategy, which partitions finely only in regions where it is needed. By limiting partitioning to losing cells only, we hope to increase the resolution in even fewer parts of the state space and thereby make the algorithm even more efficient.

To compare the performance of parti-game to the modified algorithm, we applied both algorithms to the set of continuous mazes shown in Figure 1. For all maze problems we used a simple local controller that can move directly toward the specified target state. We also

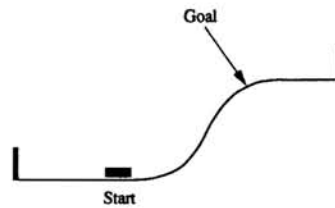

Figure 2: An ice puck on a hill. The puck can thrust horizontally to the left and to the right with a maximum force of 1 Newton. The state space is two-dimensional consisting of the horizontal position and velocity. The agent starts at the position marked Start at velocity zero and its goal is to reach the position marked Goal at velocity zero. Maximum thrust is not adequate to get the puck up the ramp so it has to learn to move to the left first to build up momentum.

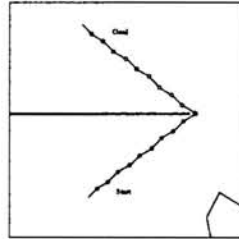

Figure 3: A nine degree of freedom, snake-like arm that moves in a plane and is fixed at one tip, as depicted in Figure 3. The objective is to move the arm from the start configuration to the goal one, which requires curling and uncurling to avoid the barrier and the wall.

applied both algorithms to the non-linear dynamics problem of the ice puck on a hill, depicted in Figure 2, which has been studied extensively in reinforcement learning literature. We used a local controller very similar to the one described in Moore and Atkeson (1995). Finally, we applied the algorithm to the nine-degree of freedom planar robot introduced in Moore and Atkeson (1995) and shown in Figure 3 and we used the same local controller described there. Additional results on the Acrobot problem (Sutton and Barto 1998) were not included here for space limitations but can be found in Al-Ansari and Williams (1998).

We applied both algorithms to each of these problems, in each case performing as many trials as was needed for the solution to stabilize. The agent was placed back in the start state at the end of each trial. In the puck problem, the agent was also reset to the start state whenever it hit either of the barriers at the bottom and top of the slope. The results are shown in Table 1. The table compares the number of trials needed, the number of partitions, total number of steps taken in the world, and the length of the final trajectory.

The table shows that the new algorithm indeed resulted in fewer total partitions in all prob-

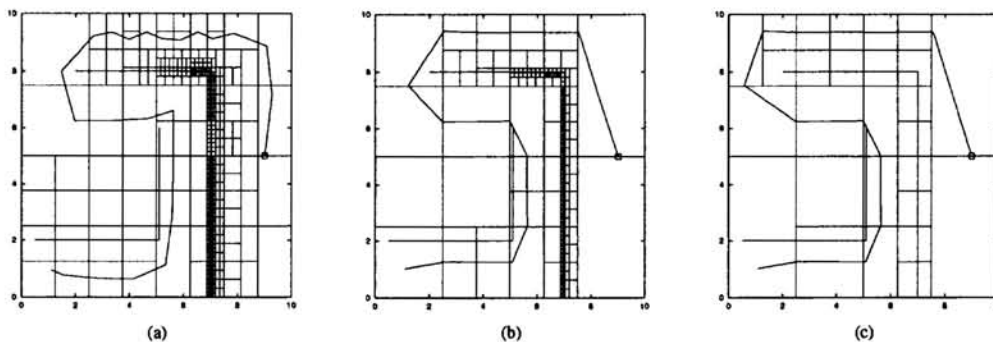

Figure 4: The final trial of applying the various algorithms to the maze in Figure 1(a). (a) parti-game, (b) parti-game with partitioning only losing cells and (c) parti-game with partitioning only the largest losing cells.

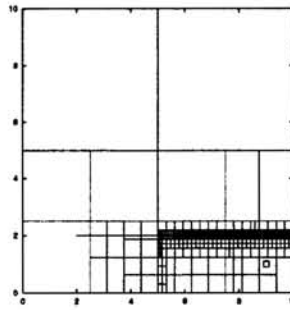

Figure 5: Parti-game needed 1194 partitions to reach the goal in the maze of Figure 1(d).

lems. It also improved in all problems in the number of trials required to stabilization. It improved in all but one problem (maze d) in the length of the final trajectory, however the difference in length is very small. Finally, it resulted in fewer total steps taken in three of the six problems, but the total steps taken increased in the remaining three.

To see the effect of the modification in detail, we show the result of applying parti-game and the modified algorithm on the maze of Figure 1(a) in Figures 4(a) and 4(b), respectively. We can see how areas with higher resolution are more localized in Figure 4(b).

## 3   BALANCED PARTITIONING

Upon close observation of Figure 4(a), we see that parti-game partitions very finely along the right wall of the maze. This behavior is even more clearly seen in parti-game's solution to the maze in Figure 1(d), which is a simple maze with a single barrier between the start state and the goal. As we see in Table 1, parti-game has a very hard time reaching the goal in this maze. Figure 5 shows the 1194 partitions that parti-game generated in trying to reach the goal. We can see that partitioning along the barrier is very uneven, being extremely fine near the goal and growing coarser as the distance from the goal increases. Putting higher focus on places where the highest gain could be attained if a hole is found can be a desirable feature, but what happens in cases like this one is obviously excessive.

One of the factors contributing to this problem of continuing to search at ever-higher resolutions in the part of the barrier nearest the goal is that any version of parti-game searches for solutions using an implicit trade-off between the shortness of a potential solution path and the resolution required to find this path. Only when the resolution becomes so fine that the number of cells through which the agent would have to pass in this potential shortcut exceeds the number of cells to be traversed when traveling around the barrier is the algorithm forced to look elsewhere for the actual opening.

A conceptually appealing way to bias this search is to maintain a more explicit coarse-to-fine search strategy. One way to do this is to try to keep the smallest cell size the algorithm generates as large as possible. In addition to achieving the balance we are seeking, this would tend to lower the total number of partitions and result in shallower tree structures needed to represent the state space, which, in turn, results in higher efficiency.

To achieve these goals, we modified the algorithm from the previous section such that whenever partitioning is required, instead of partitioning all losing cells, we only partition those among them that are of maximum size. This has the effect of postponing splits that would lower the minimum cell size as long as possible. The results of applying the modified algorithm on the test problems are also shown in Table 1.

Comparing the results of this version of the algorithm to those of partitioning all losing cells

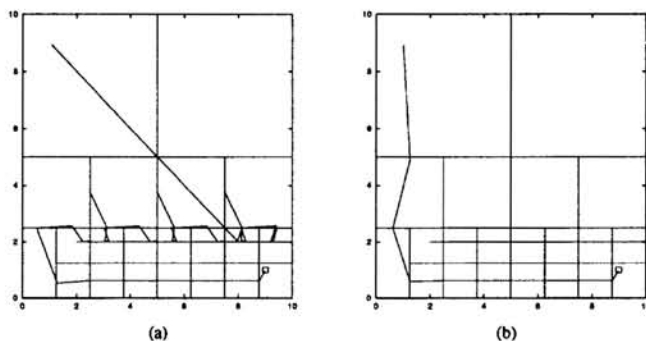

<center>(a)     (b)</center>

Figure 6: The result of partitioning largest cells on the losing side in the maze of Figure 1(d). Only two trials are required to stabilize. The first requires 1304 steps and 21 partitions. The second trial adds no new partitions and produces a path of only 165 steps.

| Problem | Algorithm | Trials | Partitions | Total Steps | Final Trajectory Length |
|---|---|---|---|---|---|
| maze a | original parti-game | 3 | 444 | 35131 | 279 |
|  | partition losing side | 3 | 239 | 16652 | **256** |
|  | partition largest losing | 3 | **27** | **1977** | 270 |
| maze b | original parti-game | 6 | 98 | **5180** | 183 |
|  | partition losing side | **5** | 76 | 7187 | 175 |
|  | partition largest losing | 6 | 76 | 5635 | **174** |
| maze c | original parti-game | 3 | 176 | 7768 | 416 |
|  | partition losing side | 2 | 120 | 10429 | 165 |
|  | partition largest losing | 2 | **96** | **6803** | 165 |
| maze d | original parti-game | 2 | 1194 | 553340 | **149** |
|  | partition losing side | 2 | 350 | 18639 | 155 |
|  | partition largest losing | 2 | **21** | **1469** | 165 |
| puck | original parti-game | 6 | 80 | 6764 | 240 |
|  | partition losing side | 2 | **18** | **3237** | **151** |
|  | partition largest losing | 2 | **18** | **3237** | **151** |
| nine-joint arm | original parti-game | 25 | 104 | 2970 | 58 |
|  | partition losing side | 17 | 61 | 3041 | **56** |
|  | partition largest losing | **7** | **37** | **2694** | 112 |

Table 1: Results of applying parti-game, parti-game with partitioning only losing cells and parti-game with partitioning the largest losing cells on three of the problem domains. Smaller numbers are better. Best numbers are shown in **bold**.

on the win-lose boundary shows that this algorithm improves on parti-game's performance even further. It outperforms the above algorithm in four problems in the total number of partitions required, while it ties it in the remaining two. It outperforms the above algorithm in total steps taken in five problems and ties it in one. It improves in the number of trials needed to stabilize in one problem, ties the above algorithm in four cases and ties parti-game in the remaining one. In the length of the final trajectory, partitioning the largest losing cells does better in one case, ties partitioning only losing cells in two cases and does worse in three. This latter result is due to the generally larger partition sizes that result from the lower resolution that this algorithm produces. However, the increase in the number of steps is very minimal in all but the nine-joint arm problem.

Figure 4(c) shows the result of applying the new algorithm to the maze of Figure 1(a). In contrast to the other two algorithms depicted in the same figure, we can see that the new algorithm partitions very uniformly around the barrier. In addition, it requires the fewest number of partitions and total steps out of the three algorithms. Figure 6 shows that the new algorithm vastly outperforms parti-game on the maze in Figure 1(d). Here, too, it partitions very evenly around the barrier and finds the goal very quickly, requiring far fewer steps and partitions.

# 4   GLOBAL PATH IMPROVEMENT

Parti-game does not claim to find optimal solutions. As we see in Figure 4, parti-game and the two modified algorithms settle on the longer of the two possible routes to the goal in this maze. In this section we investigate ways we could improve parti-game so that it could find paths of optimal form. It is important to note that we are not seeking paths that are optimal, since that is not possible to achieve using the cell shapes and aiming strategies we are using here. By a path of optimal form we mean a path that could be continuously deformed into an optimal path.

## 4.1   OTHER GRADIENTS

As mentioned above, parti-game partitions only when the agent has no winning cells to aim for and the only cells partitioned are those that lie on the win-lose boundary. The win-lose boundary falls on the gradient between finite- and infinite-cost cells and it appears when the algorithm knows of no reliable way to get to the goal. Consistently partitioning along this gradient guarantees that the algorithm will eventually find a path to the goal, if one exists.

However, gradients across which the difference in cost is finite also exist in a state space partitioned by parti-game (or any of the variants introduced in this paper). Like the win-lose boundary, these gradients are boundaries through which the agent does not believe it can move directly. Although finding an opening in such a boundary is not essential to reaching the goal, these boundaries do represent potential shortcuts that might improve the agent's policy. Any gradient with a difference in cost of two or more is a location of such a potentially useful shortcut.

Because such gradients appear throughout the space, we need to be selective about which ones to partition along. There are many possible strategies one might consider using to incorporate these ideas into parti-game. For example, since parti-game focuses on the highest gradients only, the first thing that comes to mind is to follow in parti-game's footsteps and assign partitioning priorities to cells along gradients based on the differences in values across those gradients. However, since the true cost function typically has discontinuities, it is clear that the effect of such a strategy would be to continue refining the partitioning indefinitely along such a discontinuity in a vain search for a nonexistent shortcut.

## 4.2   THE ALGORITHM

A much better idea is to try to pick cells to partition in a way that would achieve balanced partitioning, following the rationale we introduced in section 3. Again, such a strategy would result in a uniform coarse-to-fine search for better paths along those other gradients.

The following discussion could, in principle, apply to any of the three forms of parti-game studied up to this point. Because of the superior behavior of the version where we partition the largest cells on the losing side, this is the specific version we report on here, and we use the term *modified parti-game* to refer to it.

The way we incorporated partitioning along other gradients is as follows. At the end of any trial in which the agent is able to go from the start state to the goal without any unexpected results of any of its aiming attempts, we partition the largest "losing cells" (i.e., higher-cost cells) that fall on any gradient across which costs differ by more than one. Because data about experiences involving cells that are partitioned is discarded, the next time modified parti-game is run, the agent will try to go through the newly formed cells in search of a shortcut.

This algorithm amounts to simply running modified parti-game until a stable solution is

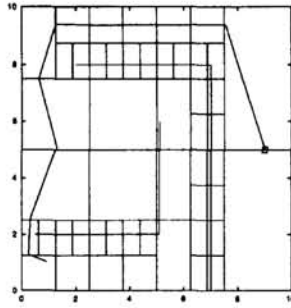

Figure 7: The solution found by applying the global improvement algorithm on the maze of Figure 1(a). The solution proceeded exactly like that of the algorithm of section 3 until the solution in Figure 4(d) was reached. After that, eight additional iterations were needed to find the better trajectory, resulting in 22 additional partitions, for a total of 49.

reached. At that point, it introduces new cells along some of the other gradients, and when it is subsequently run, modified parti-game is applied again until stabilization is achieved, and so on. The results of applying this algorithm to the maze of Figure 1(a) is shown in Figure 7. As we can see, the algorithm finds the better solution by increasing the resolution around the relevant part of the barrier above the start state.

In the absence of information about the form of the optimal trajectory, there is no natural termination criterion for this algorithm. It is designed to be run continually in search of better solutions. If, however, the form of the optimal solution is known in advance, the extra partitioning could be turned off after such a solution is found.

## 5   CONCLUSIONS

In this paper we have presented three successive modifications to parti-game. The combination of the first two appears to improve its robustness and efficiency, sometimes dramatically, and generally yields better solutions. The third provides a novel way of performing non-local search for higher quality solutions that are closer to optimal.

## Acknowledgments

Mohammad Al-Ansari acknowledges the continued support of King Saud University, Riyadh, Saudi Arabia and the Saudi Arabian Cultural Mission to the U.S.A.

## References

Al-Ansari, M. A. and R. J. Williams (1998). Modifying the parti-game algorithm for increased robustness, higher efficiency and better policies. Technical Report NU-CCS-98-13, College of Computer Science, Northeastern University, Boston, MA.

Moore, A. (1994a). Variable resolution reinforcement learning. In *Proceedings of the Eighth Yale Workshop on Adaptive and Learning Systems*. Center for Systems Science, Yale University.

Moore, A. W. (1994b). The parti-game algorithm for variable resolution reinforcement learning in multidimensional state spaces. In *Proceedings of Neural Information Processing Systems Conference 6*. Morgan Kaufman.

Moore, A. W. and C. G. Atkeson (1995). The parti-game algorithm for variable resolution reinforcement learning in multidimensional state-spaces. *Machine Learning 21*.

Sutton, R. S. and A. G. Barto (1998). *Reinforcement Learning: An Introduction*. MIT Press.